# MAP estimation in Binary MRFs via Bipartite Multi-cuts

**Sashank J. Reddi***
IIT Bombay
sashank@cse.iitb.ac.in

**Sunita Sarawagi**
IIT Bombay
sunita@cse.iitb.ac.in

**Sundar Vishwanathan**
IIT Bombay
sundar@cse.iitb.ac.in

## Abstract

We propose a new LP relaxation for obtaining the MAP assignment of a binary MRF with pairwise potentials. Our relaxation is derived from reducing the MAP assignment problem to an instance of a recently proposed Bipartite Multi-cut problem where the LP relaxation is guaranteed to provide an $O(\log k)$ approximation where $k$ is the number of vertices adjacent to non-submodular edges in the MRF. We then propose a combinatorial algorithm to efficiently solve the LP and also provide a lower bound by concurrently solving its dual to within an $\epsilon$ approximation. The algorithm is up to an order of magnitude faster and provides better MAP scores and bounds than the state of the art message passing algorithm of [1] that tightens the local marginal polytope with third-order marginal constraints.

## 1 Introduction

We consider pairwise Markov Random Field (MRF) over $n$ binary variables $\mathbf{x} = x_1, \ldots, x_n$ expressed as a graph $G = (\mathcal{V}, \mathcal{E})$ and an energy function $E(\mathbf{x}|\theta)$ whose parameters $\theta$ decompose over its vertices and edges as:

$$E(\mathbf{x}|\theta) = \sum_{i \in \mathcal{V}} \theta_i(x_i) + \sum_{(i,j) \in \mathcal{E}} \theta_{ij}(x_i, x_j) + \theta_{const} \qquad (1)$$

Our goal is to find a $\mathbf{x}^* = \operatorname{argmin}_{\mathbf{x} \in \{0,1\}^n} E(\mathbf{x}|\theta)$. This is called the MAP assignment problem in graphical models and for general graphs and arbitrary parameters is NP complete. Consequently, there is an extensive literature of approximation schemes for the problem and new algorithms continue to be explored [2, 3, 4, 5, 6, 7, 8]. The most popular of these are based on the following linear programming relaxation of the MAP problem.

$$
\begin{aligned}
\min_{\mu} &\sum_{i,x_i} \theta_i(x_i)\mu_i(x_i) + \sum_{(i,j),x_i,x_j} \theta_{ij}(x_i, x_j)\mu_{ij}(x_i, x_j) \\
&\sum_{x_j} \mu_{ij}(x_i, x_j) = \mu_i(x_i) \quad \forall (i,j) \in \mathcal{E}, \forall x_i \in \{0,1\} \\
&\sum_{x_i} \mu_i(x_i) = 1 \quad \forall i \in \mathcal{V}, \ \mu_{ij}(x_i, x_j) \geq 0 \quad \forall (i,j) \in \mathcal{E}, \forall x_i, x_j \in \{0,1\}
\end{aligned}
\qquad (2)
$$

Broadly two main techniques are used to solve this relaxation: message-passing algorithms [9, 10, 11, 7, 12] such as TRW-S and Max-sum diffusion on the dual and, combinatorial algorithms based on graph cuts and network flows [13, 14]. Both these methods find the exact MAP when the edge parameters are submodular. For non-submodular parameters, these methods provide partial optimality guarantees for variables that get integral values. This observation is exploited in [14] to design

an iterative probing scheme to expand the set of variables with optimal assignments. However, this scheme is useful only for the case when the graphical model has a few non-submodular edges. More principled methods to improve the solution output by the relaxed LP are based on progressively tightening the relaxation with violated constraints. Cycle constraints [15, 16, 17, 18, 1, 19] and higher order marginal constraints [17, 1, 20] are two such types of constraints. However, these are not backed by efficient algorithms and thus most of these tightenings come at a considerable computational cost.

In this paper we propose a new relaxation of the MAP estimation problem via reduction to a recently proposed Bipartite Multi-cut problem in undirected graphs [21]. We exploit this to show that after adding a polynomial number of constraints, we get a $O(\log k)$ approximation guarantee on the MAP objective where $k$ is the number of variables adjacent to non-submodular edges in the graphical model, and this can be tightened to $O(\sqrt{\log(k)} \log(\log(k)))$ using a semi-definite programming relaxation[1]. In this paper we explore only LP-based relaxation since our goal is to design practical algorithms.

We propose a combinatorial algorithm to efficiently solve this LP by casting it as a Multi-cut problem on a specially constructed graph, the dual of which is a multi-commodity flow problem. The algorithm, adapted from [22, 23], simultaneously updates the primal and dual solutions, and thus at any point provides both a candidate solution and a lower bound to the energy function. It is guaranteed to provide an $\epsilon$- approximate solution of the primal LP in $O(\epsilon^{-2}(|\mathcal{V}|+|\mathcal{E}|)^2)$ time but in practice terminates much faster. No such guarantees exist for any of the existing algorithms for tightening the MAP LP based on cycle or higher order marginals constraints. Empirically, this algorithm is an order of magnitude faster than the state of the art message passing algorithm[1] while yielding the same or better MAP values and bounds. We show that our LP is a relaxation of the LP with cycle constraints, but we still yield better and faster bounds because our combinatorial algorithm solves the LP within a guaranteed $\epsilon$ approximation.

## 2   MAP estimation as Bipartite Multi-cut

We assume a reparameterization of the energy function so that the parameters of $E(\mathbf{x}|\theta)$ (Equation 1) are

1. Symmetric, that is for $\{x_i,\ x_j\} \in \{0,1\}^2$ $\theta_{ij}(x_i,x_j) = \theta_{ij}(\overline{x}_i,\overline{x}_j)$ where $\overline{x}_i = 1 - x_i$,
2. Zero-normalized, that is $\min\limits_{x_i} \theta_i(x_i) = 0$ and $\min\limits_{x_i,x_j} \theta_{ij}(x_i,x_j) = 0$.

It is easy to see that any energy function over binary variables can be reparameterized in this form[2].

Our starting point is the LP relaxation proposed in [13] for approximating MAP $\mathbf{x}^* = \mathrm{argmin}_{\mathbf{x}} E(\mathbf{x}|\theta)$ as the minimum s-t cut in a suitably constructed graph $H = (\mathcal{V}_H, \mathcal{E}_H)$. We present this construction for completeness.

### 2.1   Graph cut-based relaxation of [13]

For ease of notation, first augment the $n$ variables with a special "0" variable that always takes a label of 0 and has an edge to all $n$ variables. This enables us to redefine the node parameters $\theta_i(x_i)$ as edge parameters $\theta_{0i}(0,x_i)$. Add to $H$ two vertices $i_0$ and $i_1$ for each variable $i, 0 \leq i \leq n$. For each edge $(i,j) \in \mathcal{E}$, add an edge between $i_0$ and $j_0$ with weight $\theta_{ij}(0,1)$ if the edge is submodular, else add edge $(i_0,j_1)$ with weight $\theta_{ij}(0,0)$. For every vertex $i$, if $\theta_i(1)$ is non-zero add an edge between $0_0$ and $i_0$ with weight $\theta_i(1)$ else add edge between $0_1$ and $i_0$ with weight $\theta_i(0)$. It is easy to see that the MAP problem $\min_{\mathbf{x} \in \{0,1\}^n} E(\mathbf{x})$ is equivalent to solving the following program if all

variables are further constrained to take integral values (with $D(i_0) \equiv x_i$).

$$
\begin{aligned}
\min_{d_e, D(.)} & \sum_{e \in \mathcal{E}_H} w_e d_e \\
d_e + D(i_s) - D(j_t) \geq 0 & \qquad \forall e = (i_s, j_t) \in \mathcal{E}_H \\
d_e + D(j_t) - D(i_s) \geq 0 & \qquad \forall e = (i_s, j_t) \in \mathcal{E}_H \\
D(0_0) = 0 & \qquad \qquad \text{(Min-cut LP)} \\
D(i_s) \in [0, 1] & \qquad \forall i_s \in \mathcal{V}_H \\
d_e \in [0, 1] & \qquad \forall e \in \mathcal{E}_H \\
D(i_0) + D(i_1) = 1 & \qquad \forall i \in \{0, \dots, n\}
\end{aligned}
$$

An efficient way to solve this LP exactly is by finding a s-t Min-cut in $H$ with $(s\ t)$ as $(0_0, 0_1)$ and setting $D(i_0) = 1/2$ when both $i_0$ and $i_1$ fall on the same side otherwise setting it to 0 or 1 depending on whether $i_0$ or $i_1$ are in the $0_0$ side [13, 14]. It is easy to see that this LP is equivalent to the basic LP relaxation in Equation 2 for which many alternative algorithms have been proposed [3, 6, 7, 9, 11]. On graphs with many cycles containing an odd number of non-submodular edges, this method yields poor MAP assignments.

We next show how to tighten this LP based on a connection to a recently proposed Bipartite Multi-cut problem [21].

## 2.2 Bipartite Multi-cut based LP relaxation

The Bipartite Multi-cut (BMC) problem is a generalization of the standard s-t Min-cut problem. Given an undirected graph $J = (\mathcal{N}, \mathcal{A})$ with non-negative edge weights, the s-t Min-cut problem finds the subset of edges with minimum total weight, whose deletion disconnects s and t. In BMC, we are given $k$ source-sink pairs $\text{ST} = \{(s_1, t_1) \dots (s_k, t_k)\}$, and the goal is find a subset of vertices $M \subset \mathcal{N}$ such that $|\{s_i, t_i\} \cap M| = 1$ and the total weight of edges from $M$ to the remaining vertices $\mathcal{N} - M$ is minimized. The BMC problem was recently proposed in [21] where it was shown to be NP-hard and $O(\log k)$ approximable using a linear programming relaxation. The BMC problem is also related to the more popular Multi-cut problem where the goal is to identify the smallest weight set of edges such that every $s_i$ and $t_i$ are separated. Any feasible BMC solution is a solution to Multi-cut but not the other way round. To see this, consider a graph over six vertices $(s_1, s_2, s_3, t_1, t_2, t_3)$ and three edges $(s_1, s_3), (t_1, t_2), (s_2, t_3)$. If $\text{ST} = \{(s_i, t_i) : 1 \leq i \leq 3\}$, then all pairs in ST are separated and optimal Multi-cut solution has cost 0. But, for BMC one of the three edges has to be cut. The LP relaxations for Multi-cut provide only a $\Omega(k)$ approximation to the BMC problem.

We reduce the MAP estimation problem to the Bipartite Multi-cut problem on an optimized version of graph $H$ constructed so that the set of variables $R$ adjacent to non-submodular edges is minimized. Later in Section 2.3 we will show how to create such an optimized graph. Without loss of generality, we assume that the variables in $R$ are $0, 1, \dots, k$. The remaining variables $j \in \mathcal{V} - R$ do not need the $j_1$ copy of $j$ in $H$ since there have no edges adjacent to $j_1$. We create an instance of a Bipartite Multi-cut problem on $H$ with the source-sink pairs $\text{ST} = \{(i_0, i_1) : 0 \leq i \leq k\}$. Let $M$ be the subset of vertices output by BMC on this graph, and without loss of generality assume that $M$ contains $0_0$. The MAP labeling $\mathbf{x}^*$ is obtained from $M$ by setting $x_i = s$ if $i_s \in M$ and $x_i = \bar{s}$ if $i_s \in \mathcal{V}_H - M$. This gives a valid MAP labeling because for each variable $j$ that appears in the set $R$, BMC ensures that $M$ contains exactly one of $(j_0, j_1)$.

Using this connection, we tighten the Min-cut LP as follows. For each $u \in \{0_0, 0_1, \dots, k_0, k_1\}$ and $j_s \in \mathcal{V}_H$ we define new variables $D_u(j_s)$ and use these to augment the Min-cut LP with additional

constraints as follows:

$$\min_{d_e, D_u(.)} \sum_{e \in \mathcal{E}_H} w_e d_e$$

$$\left.\begin{array}{l} d_e + D_u(i_s) - D_u(j_t) \geq 0 \\ d_e + D_u(j_t) - D_u(i_s) \geq 0 \end{array}\right\} \ \forall e = (i_s, j_t) \in \mathcal{E}_H, \ \ \forall u \in \{0_0, 0_1 \dots, k_0, k_1\}$$

$$D_{i_0}(i_1) \geq 1 \ \ \forall i \in \{0, \dots, k\} \qquad \text{(BMC LP)}$$

$$D_u(j_s) \geq 0 \ \ \forall j_s \in \mathcal{V}_H, \ \ \forall u \in \{0_0, 0_1 \dots k_0, k_1\}$$

$$d_e \geq 0 \ \ \forall e \in \mathcal{E}_H$$

$$\left.\begin{array}{l} D_{i_0}(j_0) = D_{i_1}(j_1) \\ D_{i_0}(j_1) = D_{i_1}(j_0) \end{array}\right\} \forall i, j \in \{0, \dots, k\}$$

A useful interpretation of the above LP is provided by viewing variables $d_e$ as the distance between $i_s$ and $j_t$ for any edge $e = (i_s, j_t)$, and variables $D_u(j_s)$ as the distance between $u$ and $j_s$. The first two constraints ensure that these distance variables satisfy triangle inequality. These, along with the constraint $D_{i_0}(i_1) \geq 1$ ensure that for every ST pair $(i_0, i_1)$, any path $P$ from $i_0$ to $i_1$ has $\sum_{e \in P} d_e \geq 1$. In contrast, the Min-cut LP ensures this kind of separation only for the $(0_0, 0_1)$ terminal pair. Later, in Section 5 we will establish a connection between these constraints and cycle constraints [15, 16, 17, 18, 19]. When the LP returns integral solutions, we obtain an optimal MAP labeling using $M = \{j_s : D_{0_0}(j_s) = 0\}$. When the variables are not integral, [21] suggests a region growing approach for rounding them so as to get a $O(\log k)$ approximation of the optimal objective. In practice, we found that ICM starting with fractional node assignments $x_i = D_{0_0}(i_0)$ gave better results.

## 2.3 Reducing the size of ST set

In the LP above, for every edge that is non-submodular we add a terminal pair to ST corresponding to any of its two endpoints. The problem of minimizing the size of the ST set is equivalent to the problem of finding the minimum set $R$ of variables of $G$ such that all cycles with an odd number of non-submodular edges are covered. It is easy that see that in any such cycle, it is always possible to flip the variables such that any one selected edge is non-submodular and the rest are submodular. Since finding the optimal $R$ is NP-hard, we used the following heuristics.

First, we pick the set of variables to flip so as to minimize the number of non-submodular edges, and then obtain a vertex cover of the reduced non-submodular edges using a greedy algorithm. Interestingly, this problem can be cast as a MAP inference problem on $G$ defined as follows: For each variable, label 0 denotes that the variable is not flipped and 1 denotes that the node is flipped. Thus, if an edge is submodular and both variables attached to it are flipped (i.e labeled 1) then the edge remains submodular. We need to minimize the number of non-submodular edges. Therefore, energy function for this new graphical model will be

$$\bar{\theta}_{ij}(x_i, x_j) = x_i \oplus x_j \oplus \text{is\_non\_submodular}(i, j) \quad \forall (i, j) \in \mathcal{E}$$

$$\bar{\theta}_i(0) = \bar{\theta}_i(1) = 0 \quad \forall i \in \mathcal{V}$$

When $G$ is planar, for example a grid, the special structure of these potentials (Ising energy function) enables us to get an optimal solution using the matching algorithm of [24, 8].

With the above LP formulation, we were able to obtain exact solutions for most 20x20 grids and 25 node clique graphs. However, the LP does not scale beyond 30x30 grid and 50 node clique graphs. We therefore provide a combinatorial algorithm for solving the LP.

## 3 Combinatorial algorithm

We will adapt the primal-dual algorithm that was proposed in [22, 23] for solving the closely related Multi-cut problem. We review this algorithm in Section 3.1 and in Section 3.2 show how we adapt it to solve the BMC LP.

## 3.1 Garg's algorithm for the Multi-cut problem

Recall that in the Multi-cut problem, the goal is to remove the minimum weight set of edges so as to separate each $(s_i, t_i)$ pair in ST. This problem is formulated as the followed primal dual LP pair in [22].

<div>

Multi-cut LP: Primal

$$\min_d \sum_{e \in \mathcal{E}_H} w_e d_e$$

$$\sum_{e \in P} d_e \geq 1 \quad \forall P \in \mathcal{P}$$

$$d_e \geq 0 \quad \forall e \in \mathcal{E}_H$$

Multi-cut LP: Dual

$$\max_f \sum_{P \in \mathcal{P}} f_P$$

$$\sum_{P \in \mathcal{P}_e} f_P \leq w_e \quad \forall e \in E_H$$

$$f_P \geq 0 \quad \forall P \in \mathcal{P}$$

</div>

where $\mathcal{P}$ denotes all paths between a pair of vertices in ST and $\mathcal{P}_e$ denotes the set of paths in $\mathcal{P}$ which contain edge $e$. Garg's algorithm [22, 23] simultaneously solves the primal and dual so that they are within an $\epsilon$ factor of each other for any user-provided $\epsilon > 0$. The algorithm starts by setting all dual variables flow variables to zero and all primal variables $d_e = \delta$ where $\delta$ is $(1+\epsilon)/((1+\epsilon)L)^{1/\epsilon}$, and $L$ is the maximum number of edges for any path in $\mathcal{P}$. It then iteratively updates the variables by first finding the shortest path $P \in \mathcal{P}$ which violates the $\sum_{e \in P} d_e \geq 1$ constraint and then, modifying variables as $f_P = \min_{e \in P} w_e$ i.e $f = f + f_P$ and $d_e = d_e(1 + \frac{\epsilon f_P}{w_e}) \; \forall e \in P$. At any point a feasible solution can be obtained by rescaling all the primal and dual variables. Termination is reached when the rescaled primal objective is within $(1 + \epsilon)$ of the rescaled dual objective for error parameter $\epsilon$. This process is shown to terminate in $O(m \, log_{1+\epsilon} \frac{1+\epsilon}{\delta})$ steps where $m = |\mathcal{E}_H|$.

## 3.2 Solving the BMC LP

We first modify the edge weights on graph $H$ constructed for the BMC LP so that for all edges $e = (i_s, j_t)$ and its complement $\bar{e} = (i_{\bar{s}}, j_{\bar{t}})$, the weights are equal, that is, $w_e = w_{\bar{e}}$. This can be easily ensured by setting $w_e = w_{\bar{e}} =$ average of previous edge weights of $e$ and $\bar{e}$ in $H$. This change adds all $(2n + 2)$ possible vertices to $H$ i.e all nodes $0 \leq i \leq n$ contain terminal pairs $(i_0, i_1)$ in the ST set. For any path $P$ in $H$ we define its *complementary* path $\bar{P}$ to be the path obtained by reversing the order of edges and complementing all edges in $P$. For example, the complement of path $(2_0, 1_1, 3_0, 2_1)$ is $(2_0, 3_1, 1_0, 2_1)$. Next, we consider the following alternative LP called BMC-Sym LP for BMC on symmetric graphs, that is, graphs where $w_e = w_{\bar{e}}$

$$\min \sum_{e \in \mathcal{E}_H} w_e d_e$$

$$\sum_{e \in P} d_e \geq 1 \quad \forall P \in \mathcal{P} \qquad \text{(BMC-Sym LP)}$$

$$d_e \geq 0, \; d_e = d_{\bar{e}} \quad \forall e \in \mathcal{E}_H$$

**Lemma 1** *When $H$ is symmetric, the BMC-Sym LP, BMC LP, and Multi-cut LP are equivalent.*

PROOF Any feasible solution of BMC-Sym LP can be used to obtain a solution to BMC LP with the same objective as follows: Set $d_e$ variables unchanged, this keeps the objective intact. Set $D_u(i_s)$ as the length of the shortest path between $u$ and $i_s$ that is, $D_u(i_s) = \min_{P \in \text{paths}(u, i_s)} \sum_{e \in P} d_e$. This yields a feasible solution — the constraints $d_e + D_u(i_s) - D_u(j_t) \geq 0$ hold because $D_u(i_s)$ variables are the shortest path between $u$ and $i_s$. The constraints $D_{i_0}(i_1) \geq 1$ hold because all paths between $i_0$ and $i_1$ have a distance $\geq 1$ in BMC-Sym LP. The constraints $D_{i_0}(j_0) = D_{i_1}(j_1)$ and $D_{i_0}(j_1) = D_{i_1}(j_0)$ are satisfied because the distances are symmetric $d_e = d_{\bar{e}}$.

We next show that any feasible solution of BMC LP gives a feasible solution to Multi-cut LP with the same $d_e$ and objective value. For any pair $(p_0, p_1) \in ST$ the constraint $D_{p_0}(p_1) \geq 1$ along with repeated application of $d_e + D_{p_0}(i_s) - D_{p_0}(j_t) \geq 0$ ensures that $\sum_{e \in P} d_e \geq 1$ for any path between $p_0$ and $p_1$.

Finally, we show that if $\{d_e\}$ is a feasible solution to Multi-cut LP then it can be used to construct a feasible solution $\{d'_e\}$ to BMC-Sym LP without changing the value of the objective function using

$d'_e = d'_{\overline{e}} = (d_e + d_{\overline{e}})/2$. The objective value remains unchanged since $w_e = w_{\overline{e}}$. The path constraints $\sum_{e \in P} d'_e \geq 1$ hold $\forall P \in \mathcal{P}$ because both path $P$ and its complementary path $\overline{P}$ are in $\mathcal{P}$ and we know that $\sum_{e \in P} d_e \geq 1$ and $\sum_{e \in \overline{P}} d_e = \sum_{e \in P} d_{\overline{e}} \geq 1$.

We modify Garg's algorithm [22, 23] to exploit the fact that the graph is symmetric so that at each iteration we push twice the flow while keeping the approximation guarantees intact. The key change we make is that when augmenting flow $f$ in some path $P$, we augment the same flow $f$ to the complementary path $\overline{P}$ as outlined in our final algorithm in Figure 1. This change ensures that we always obtain symmetric distance values as we prove below.

**Lemma 2** *Suppose $H$ is a symmetric graph then $d_e = d_{\overline{e}} \forall e \in \mathcal{E}_H$ at the end of each iteration of the while loop in algorithm in Figure 1.*

PROOF We prove by induction. The claim holds initially, since $d_e = \delta \ \forall e \in \mathcal{E}_H$ and $H$ is symmetric. Let $P_i$ denote the path selected in the $i^{th}$ iteration of the algorithm. Now, suppose that the hypothesis is true for the $n^{th}$ iteration. In the $(n+1)^{th}$ iteration, we augment flow $f$ in both paths $P_{n+1}$ and $\overline{P}_{n+1}$. These paths $P_{n+1}$ and $\overline{P}_{n+1}$ do not share any edge because this would imply that there is another pair $(j_0, j_1)$ of shorter length, and we would choose $P_{n+1}$ to be this path instead. We then do the following update $d_e = d_e(1 + \frac{\epsilon f_P}{w_e})$ with $f_P = \min_{e \in P} w_e$ for both the paths $P_{n+1}$ and $\overline{P}_{n+1}$. Since $w_e = w_{\overline{e}}$ for all $e \in \mathcal{E}$ and $d_e = d_{\overline{e}} \forall e \in \mathcal{E}_H$ before this iteration, $d_e = d_{\overline{e}}$ $\forall e \in \mathcal{E}_H$ after $(n+1)^{th}$ step.

**Theorem 3** *The modified algorithm also provides an $\epsilon$-approximation algorithm to the BMC LP.*

PROOF Suppose, we do not augment the flow in the complementary path $\overline{P}$ while augmenting $P$. In the next iteration the original algorithm of [22, 23] picks $\overline{P}$ or any path with the same path length since the path length of $\overline{P}$ and $P$ is equal before the iteration and they do not share any common edges. Therefore, by forcing $\overline{P}$ we are not modifying the course of the original algorithm and the analysis in [22, 23] holds here as well.

---

**Input:** Graphical model $G$ with reparameterized energy function $E$, approximation guarantee $\epsilon$
Create symmetric graph $H$ from $G$ and $E$
Initialize $d_e = \delta$ ($\delta$ derived from $\epsilon$ as shown in Section 3.1), and $f = 0, f_e = 0$,
**x**=arbitrary initial labeling of graphical model $G$.
Define: Primal objective $P(\{d_e\}) = \sum_e w_e d_e / \min_{P \in \mathcal{P}} \sum_{e \in P} d_e$
Define: Dual objective $D(f, \{f_e\}) = f / (\max_e f_e / w_e)$
**while** $\min (E(\mathbf{x}) - \theta_{const}, P(\{d_e\})) > (1 + \epsilon) D(f, \{f_e\})$ **do**
  $P$ = Shortest path between $(i_0, i_1)$ $\forall (i_0, i_1) \in$ ST
  **if** $(\sum_{e \in P} d_e < 1)$ **then**
    With $f_P = \min_{e \in P} w_e$ update $f = f + f_P$, $f_e = f_e + f_P$, $d_e = d_e(1 + \frac{\epsilon f_P}{w_e}) \ \forall e \in P$.
    Repeat above for the complement path $\overline{P}$
    $\mathbf{x}'$ = current solution after rounding, $\mathbf{x}$ =better of $\mathbf{x}$ and $\mathbf{x}'$
  **end if**
**end while**
**Return** bound = $D(f, \{f_e\}) + \theta_{const}$, MAP = **x**.

Figure 1: Combinatorial Algorithm for MAP inference using BMC.

Our algorithm in addition to updating the primal and dual solutions at each iteration, also keeps track of the primal objective obtained with the current best rounding (**x** in Figure 1). Often, the rounded variables yielded lower primal objective values and led to early termination. The complexity of the algorithm can be shown to be $O(\epsilon^{-2} k m^2)$ ignoring the polylog($m$) factors. Fleischer [25] subsequently improved the above algorithm by reducing the complexity to $O(\epsilon^{-2} m^2)$. It is interesting to note that running time is independent of $k$. Though we have presented modification to algorithm in [22, 23], we can fit our algorithm in Fleischer's framework as well. In fact, we use Fleischer's modification for practical implementation of our algorithm.

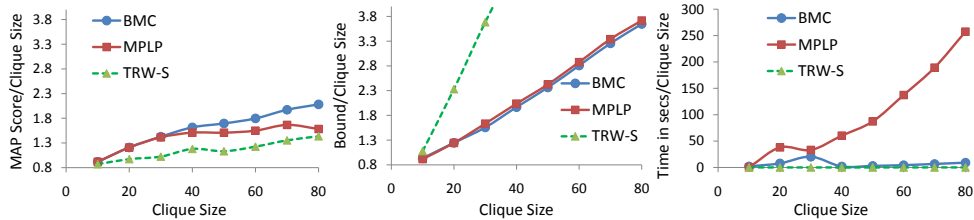

Figure 2: Clique size scaled values of MAP, Upper bound, and running time with increasing clique size on three methods: BMC, MPLP, and TRW-S.

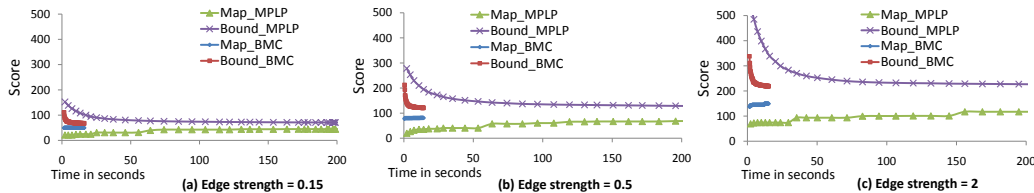

Figure 3: Comparing convergence rates of BMC and MPLP for three different clique graphs.

## 4   Experiments

We compare our proposed algorithm (called BMC here) with MPLP, a state-of-art message passing algorithm [1] that tightens the standard MAP LP with third order marginal constraints, which are equivalent to cycle constraints for binary MRFs. As reference we also present results for the TRW-S algorithm [9]. BMC is implemented in Java whereas for MPLP we ran the C++ code provided by the authors. We run BMC with $\epsilon = 0.02$. MPLP was run with edge clusters until convergence (up to a precision of $2 \times 10^{-4}$) or for at most 1000 iterations, whichever comes first. Our experiments were performed on two kinds of datasets: (1) Clique graph based binary MRFs of various sizes generated as per the method of [17] where edge potentials are Potts sampled from $U[-\sigma, \sigma]$ (our default setting was $\sigma = 0.5$) and node potentials via $U[-1, 1]$, and (2) Maxcut instances of various sizes and densities from the BiqMac library[3]. Since the second task is formulated as a maximization problem, for the sake of consistency we report all our results as maximizing the MAP score. We compare the algorithms on the quality of the final solution, the upper bound to MAP score, and running time. It should be noted that multiplicative bounds do not hold here since the reparameterizations give rise to negative constants.

In the graphs in Figure 2 we compare BMC, MPLP, and TRW-S with increasing clique size averaged over five seeds. We observe that BMC provides much higher MAP scores and slightly tighter bounds than MPLP. In terms of running time, BMC is more than an order of magnitude faster than MPLP for large graphs. The baseline LP (TRW-S) while much faster than both BMC and MPLP provides really poor MAP scores and bounds. We also compare BMC and MPLP on their speed of convergence. In Figures 3(a), (b), and (c) we show the MAP and Upper bounds for different times in the execution of the algorithm on cliques of size 50 and different edge strengths. BMC, whose bounds and MAP appear as the two short arcs in-between the MAP scores and bounds of MPLP, converges significantly faster and terminates well before MPLP while providing same or better MAP scores and bounds for all edge strengths.

In Table 1 we compare the three algorithms on the various graphs from the BiqMac library. The graphs are sorted by increasing density and are all of size 100. We observe that the MAP values for BMC are significantly higher than those for TRW-S. For MPLP, the MAP values are always zero because it decodes marginals purely based on node marginals which for these graphs are tied. The upper bounds achieved by MPLP are significantly tighter than TRW-S, showing that with proper rounding MPLP is likely to produce good MAP scores, but BMC provides even tighter bounds in

[3]http://biqmac.uni-klu.ac.at/

| Graph | density | MAP | | | Bound | | | Time in seconds | | |
|---|---|---|---|---|---|---|---|---|---|---|
| | | BMC | MPLP | TRW-S | BMC | MPLP | TRW-S | BMC | MPLP | TRW-S |
| pm1s | 0.1 | 110 | 0 | 91 | 131 | 200 | 257 | 45 | 43 | 0.005 |
| pw01 | 0.1 | 1986 | 0 | 1882 | 2079 | 2397 | 2745 | 48 | 46 | 0.006 |
| w01 | 0.1 | 653 | 0 | 495 | 720 | 1115 | 1320 | 46 | 41 | 0.004 |
| g05 | 0.5 | 1409 | 0 | 1379 | 1650 | 1720 | 2475 | 761 | 317 | 0.021 |
| pw05 | 0.5 | 7975 | 0 | 7786 | 9131 | 9195 | 13696 | 699 | 1139 | 0.021 |
| w05 | 0.5 | 1444 | 0 | 1180 | 2245 | 2488 | 6588 | 737 | 1261 | 0.021 |
| pw09 | 0.9 | 13427 | 0 | 13182 | 16493 | 16404 | 24563 | 106 | 2524 | 0.041 |
| w09 | 0.9 | 1995 | 0 | 1582 | 4073 | 4095 | 11763 | 123 | 2671 | 0.053 |
| pm1d | 0.99 | 347 | 0 | 277 | 842 | 924 | 2463 | 12 | 1307 | 0.047 |

Table 1: Comparisons on Maxcut graphs of size 100 from the BiqMac library.

most cases. The running time for BMC is significantly lower than MPLP for dense graphs but for sparse graphs (10% edges) it requires the same time as MPLP.

Thus, overall we find that BMC achieves tighter bounds and better MAP solutions at a significantly faster rate than the state-of-the-art method for tightening LPs. The gain over MPLP is highest for the case of dense graphs. For sparse graphs many algorithms work, for example recently [8, 26] reported excellent results on planar, or nearly planar graphs and [27] show that even local search works when the graph is sparse.

# 5 Discussion and Conclusion

We put our tightening of the basic MAP LP (Marginal LP in Equation 2 or the Min-cut LP) in perspective with other proposed tightenings based on cycle constraints [17, 18, 1, 19] and higher order marginal constraints [17, 1, 20]. For binary MRFs cycle constraints are equivalent to adding marginal consistency constraints among triples of variables [28]. We show the relationship between cycle constraints and our constraints. Let $S = (\mathcal{V}_S, \mathcal{E}_S)$ denote the minimum cut graph created from $G$ as shown in Section 2.1 but without the $i_1$ vertices for $(1 \leq i \leq n)$ so that weights of non-submodular edges in $S$ will be negative. The LP relaxation of MAP based on cycle constraints is defined as:

$$\min_{d} \sum_{e \in \mathcal{E}_S} w'_e d_e$$
$$\sum_{e \in F} (1 - d_e) + \sum_{e \in C \backslash F} d_e \geq 1 \qquad \forall C \in \mathcal{C}, F \subseteq C \text{ and } \mid F \mid \text{ is odd}$$
$$d_e \in [0 \dots 1] \qquad \forall e \in \mathcal{E}_S$$

where $\mathcal{C}$ denotes the set of all cycles in $S$. Suppose we construct our symmetric minimum cut graph $H$ with edges $(i_s, j_t)$ corresponding to all four possible values of $(s, t)$ for each edge $(i, j) \in \mathcal{E}$, instead of two that we currently get due to zero-normalized edge potentials. Then, BMC-Sym LP along with the constraints $d_{i_s j_t} + d_{i_s j_{\bar{t}}} = 1 \ \forall (i_s, j_t) \in \mathcal{E}_H$ is equivalent to the cycle LP above. We skip the proof due to lack of space.

Our main contribution is that by relaxing the cycle LP to the Bipartite Multi-cut LP we have been able to design a combinatorial algorithm which is guaranteed to provide an $\epsilon$ approximation to the LP in polynomial time. Since we solve the LP and its dual better than any of the earlier methods of enforcing cycle constraints, we are able to obtain tighter bounds and MAP scores at a considerable faster speed.

Future work in this area includes developing combinatorial algorithm for solving the semi-definite program in [21] and extending our approach to multi label graphical models.

**Acknowledgement** We thank Naveen Garg for helpful discussion in relating the multi-commodity flow problem with the Bipartite multi-cut problem. The second author acknowledges the generous support of Microsoft Research and IBM's Faculty award.

## Footnotes

*The author is currently affiliated with Google Inc.

[1]We note however that these multiplicative bounds may not be relevant for MAP estimation problem in graphical models where reparameterization leaves behind *negative* constants which are kept outside the LP objective.

[2]Set: $\theta'_{ij}(0,0) = \theta'_{ij}(1,1) = (\theta_{ij}(0,0) + \theta_{ij}(1,1))/2$, $\theta'_{ij}(0,1) = \theta'_{ij}(1,0) = (\theta_{ij}(0,1) + \theta_{ij}(1,0))/2$, $\theta'_i(1) = \theta_i(1) + \sum_{(i,j)\in\mathcal{E}}(\theta_{ij}(1,0) + \theta_{ij}(1,1) - \theta_{ij}(0,1) - \theta_{ij}(0,0))/2$, $\theta'_{const} = \theta_{const} + \sum_{(i,j)\in\mathcal{E}}(\theta_{ij}(0,0) - \theta_{ij}(1,1))/2$. Then zero normalize as in [9].

# References

[1] David Sontag, Talya Meltzer, Amir Globerson, Tommi Jaakkola, and Yair Weiss. Tightening LP Relaxations for MAP using Message Passing. In *UAI*, 2008.

[2] D. Koller and N. Friedman. *Probabilistic Graphical Models: Principles and Techniques*. MIT Press, 2009.

[3] M.I. Schlesinger. Syntactic analysis of two-dimensional visual signals in noisy conditions. Kybernetica, 1976.

[4] Chandra Chekuri, Sanjeev Khanna, Joseph (Seffi) Naor, and Leonid Zosin. Approximation Algorithms for the Metric Labeling Problem via a New Linear Programming Formulation. In *SODA*, 2001.

[5] Jon Kleinberg and Eva Tardos. Approximation Algorithms for Classification Problems with Pairwise Relationships: Metric Labeling and Markov Random Fields. *J. ACM*, 49(5):616–639, 2002.

[6] M. Wainwright, T. Jaakkola, and A. Willsky. MAP Estimation Via Agreement on Trees: Message-Passing and Linear Programming. *IEEETIT: IEEE Transactions on Information Theory*, 51, 2005.

[7] Tomás Werner. A Linear Programming Approach to Max-Sum Problem: A Review. *IEEE Trans. Pattern Anal. Mach. Intell.*, 29(7):1165–1179, 2007.

[8] Nic Schraudolph. Polynomial-Time Exact Inference in NP-Hard Binary MRFs via Reweighted Perfect Matching. In *AISTATS*, 2010.

[9] Vladimir Kolmogorov. Convergent Tree-Reweighted Message Passing for Energy Minimization. *IEEE Trans. Pattern Anal. Mach. Intell.*, 28(10):1568–1583, 2006.

[10] Talya Meltzer, Amir Globerson, and Yair Weiss. Convergent message passing algorithms - a unifying view. In *UAI*, 2009.

[11] Pradeep Ravikumar, Alekh Agarwal, and Martin J. Wainwright. Message-passing for Graph-structured Linear Programs: Proximal Methods and Rounding Schemes. *JMLR*, 11:1043–1080, 2010.

[12] David Sontag and Tommi Jaakkola. Tree Block Coordinate Descent for MAP in Graphical Models. In *AI-STATS*, volume 9, pages 544–551, 2009.

[13] Endre Boros and Peter L. Hammer. Pseudo-Boolean Optimization. *Discrete Applied Mathematics*, 123(1-3):155–225, 2002.

[14] Carsten Rother, Vladimir Kolmogorov, Victor S. Lempitsky, and Martin Szummer. Optimizing Binary MRFs via Extended Roof Duality. In *CVPR*, 2007.

[15] Francisco Barahona and Ali Ridha Mahjoub. On the cut polytope. *Math. Program.*, 36(2):157–173, 1986.

[16] Uri Zwick. Outward Rotations: A Tool for Rounding Solutions of Semidefinite Programming Relaxations, with Applications to MAX CUT and Other Problems. In *STOC*, 1999.

[17] David Sontag and Tommi Jaakkola. New Outer Bounds on the Marginal Polytope. In *NIPS*, 2007.

[18] M. Pawan Kumar, Vladimir Kolmogorov, and Philip H. S. Torr. An Analysis of Convex Relaxations for MAP Estimation of Discrete MRFs. *JMLR*, 10:71–106, 2009.

[19] Nikos Komodakis and Nikos Paragios. Beyond Loose LP-Relaxations: Optimizing MRFs by Repairing Cycles. In *ECCV*, 2008.

[20] Tomás Werner. High-arity interactions, polyhedral relaxations, and cutting plane algorithm for soft constraint optimisation (map-mrf). In *CVPR*, 2008.

[21] Sreyash Kenkre and Sundar Vishwanathan. Approximation algorithms for the Bipartite Multicut problem. *Information Processing Letters*, 110(8-9):282 – 287, 2010.

[22] Naveen Garg, Vijay V. Vazirani, and Mihalis Yannakakis. Approximate Max-Flow Min-(Multi)Cut Theorems and Their Applications. *SIAM J. Comput.*, 25(2):235–251, 1996.

[23] Naveen Garg and Jochen Knemann. Faster and Simpler Algorithms for Multicommodity Flow and Other Fractional Packing Problems. *SIAM J. Comput. 37(2): (2007)*, 37(2):630–652, 2007.

[24] Amir Globerson and Tommi Jaakkola. Approximate inference using planar graph decomposition. In *NIPS*, 2006.

[25] Lisa Fleischer. Approximating Fractional Multicommodity Flow Independent of the Number of Commodities. *SIAM J. Discrete Math.*, 13(4):505–520, 2000.

[26] D Batra, A C Gallagher, D Parikh, and T Chen. Beyond trees: Mrf inference via outer-planar decomposition. In *CVPR*, 2010.

[27] Kyomin Jung, Pushmeet Kohli, and Devavrat Shah. Local Rules for Global MAP: When Do They Work? In *NIPS*. 2009.

[28] David Sontag. Cutting plane algorithms for variational inference in graphical models. Master's thesis, MIT, Department of Electrical Engineering and Computer Science, 2007.

